# Cobot: A Social Reinforcement Learning Agent

**Charles Lee Isbell, Jr.**   **Christian R. Shelton**
AT&T Labs-Research    Stanford University
**Michael Kearns**   **Satinder Singh**   **Peter Stone**
University of Pennsylvania   Syntek Capital   AT&T Labs-Research

## Abstract

We report on the use of reinforcement learning with Cobot, a software agent residing in the well-known online community LambdaMOO. Our initial work on Cobot (Isbell et al.2000) provided him with the ability to collect *social statistics* and report them to users. Here we describe an application of RL allowing Cobot to take proactive actions in this complex social environment, and adapt behavior from multiple sources of human reward. After 5 months of training, and 3171 reward and punishment events from 254 different LambdaMOO users, Cobot learned nontrivial preferences for a number of users, modifing his behavior based on his current state. Here we describe LambdaMOO and the state and action spaces of Cobot, and report the statistical results of the learning experiment.

## 1   Introduction

While most applications of reinforcement learning (RL) to date have been to problems of control, game playing and optimization (Sutton and Barto1998), there has been a recent handful of applications to human-computer interaction. Such applications present a number of interesting challenges to RL methodology (such as data sparsity and inevitable violations of the Markov property). These previous studies focus on systems that encounter human users one at a time, such as spoken dialogue systems (Singh et al.2000).

In this paper, we report on an RL-based agent for LambdaMOO, a complex, open-ended, multi-user chat environment, populated by a community of human users with rich and often enduring social relationships. Our long-term goal is to build an agent who can learn to perform useful, interesting and entertaining actions in LambdaMOO on the basis of user feedback. While this is a deliberately ambitious and underspecified goal, we describe here our implementation, the empirical experiences of our agent so far, and some of the lessons we have learned about this challenging domain.

In previous work (Isbell et al.2000), we developed the software agent Cobot, who interacted in various ways with LambdaMOO users. Cobot had two primary functions. First, Cobot gathered "social statistics" (*e.g.* how frequently and in what ways users interacted with one another), and provided summaries of these statistics as a service. Second, Cobot had rudimentary chatting abilities based on the application of information retrieval methods to large documents. The original Cobot was entirely *reactive*[1], in that he[2] never initiated interaction with human users, but would only respond to their actions. As we documented in our earlier paper, Cobot proved tremendously popular with LambdaMOO users, setting the stage for our current efforts.

We modified Cobot to allow him to take certain actions (such as proposing conversation topics, introducing users, or engaging in common word play routines) under his own initiative. The hope is to build an agent that will eventually take unprompted actions that are meaningful, useful or amusing to users. Rather than hand-code complex rules specifying

when each action is appropriate (rules that would be inaccurate and quickly become stale), we wanted Cobot to *learn the individual and communal preferences* of users. Thus, we provided a mechanism for users to reward or punish Cobot, and programmed Cobot to use RL algorithms to alter his behavior on the basis of this feedback. The application of RL (or any machine learning methodology) to such an environment presents a number of interesting domain-specific challenges, including:

- **Choice of an appropriate state space.** To learn how to act in a social environment such as LambdaMOO, Cobot must represent the salient features. These should include social information such as which users are present, how experienced they are in LambdaMOO, how frequently they interact with one another, and so on.

- **Multiple reward sources.** Cobot lives in an environment with multiple, often conflicting sources of reward from different human users. How to integrate these sources reasonably is a nontrivial empirical question.

- **Inconsistency and drift of user rewards and desires.** Individual users may be inconsistent in the rewards they provide (even when they implicitly have a fixed set of preferences), and their preferences may change over time (for example, due to becoming bored or irritated with an action). Even when their rewards are consistent, there can be great temporal variation in their reward pattern.

- **Variability in user understanding.** There is great variation in users' understanding of Cobot's functionality, and the effects of their rewards and punishments.

- **Data sparsity.** Training data is scarce for many reasons, including user fickleness, and the need to prevent Cobot from generating too much spam in the environment.

- **Irreproducibility of experiments.** As LambdaMOO is a globally distributed community of human users, it is virtually impossible to replicate experiments taking place there.

We do not have any simple answers (nor do we believe that simple answers exist), but here we provide a case study of our choices and findings. Our primary findings are:

- **Inappropriateness of average reward.** We found that the average reward that Cobot received over time, the standard measure of success for RL experiments, is an inadequate and perhaps even inappropriate metric of performance in the LambdaMOO domain. Reasons include that user preferences are not stationary, but drift as users become habituated or bored with Cobot's behavior; and the tendency for satisfied users to stop providing Cobot with any feedback, positive or negative. Despite the inadequacy of average reward, we are still able to establish several measures by which Cobot's RL succeeds, discussed below.

- **A small set of dedicated "parents".** While many users provided only a moderate or small amount of RL training (rewards and punishments) to Cobot, a handful of users did invest significant time in training him.

- **Some parents have strong opinions.** While many of the users that trained Cobot did not exhibit clear preferences for any of his actions over the others, some users clearly and consistently rewarded and punished particular actions over the others.

- **Cobot learns matching policies.** For those users who exhibited clear preferences through their rewards and punishments, Cobot successfully learned corresponding policies of behavior.

- **Cobot responds to his dedicated parents.** For those users who invested the most training time in Cobot, the observed distribution of his actions is significantly altered by their presence.

- **Some preferences depend on state.** Although some users for whom we have sufficient data seem to have preferences that do not depend upon the social state features we constructed for the RL, others do in fact appear to change their preferences depending upon prevailing social conditions.

The outline for the rest of the paper is as follows. In Section 2, we give brief background on LambdaMOO. In Section 3, we describe our earlier (non-RL) work on Cobot. Section 4 provides some brief background on RL. In Sections 5, 6 and 7 we describe our implementation of Cobot's RL action space, reward mechanisms and state features, respectively. Our primary findings are presented in Section 8, and Section 9 offers conclusions.

## 2 LambdaMOO

LambdaMOO, founded in 1990 by Pavel Curtis at Xerox PARC, is the oldest continuously operating MUD, a class of online worlds with roots in text-based multiplayer role-playing games. MUDs (multi-user dungeons) differ from most chat and gaming systems in their use of a persistent representation of a virtual world, often created by the participants, who are represented as characters of their own choosing. LambdaMOO appears as a series of interconnected rooms, populated by users and objects who may move between them. Each room provides a shared chat channel, and typically has an elaborate text description that imbues it with its own "look and feel." In addition to speech, users express themselves via a large collection of *verbs*, allowing a rich set of simulated actions, and the expression of emotional states:

(1)  Buster is overwhelmed by all these deadlines.
(2)  Buster begins to slowly tear his hair out, one strand at a time.
(3)  HFh comforts Buster.
(4)  HFh [to Buster]: Remember, the mighty oak was once a nut like you.
(5)  Buster [to HFh]: Right, but his personal growth was assured. Thanks anyway, though.
(6)  Buster feels better now.

Lines (1) and (2) are initiated by verb commands by user Buster, expressing his emotional state, while lines (3) and (4) are examples of verbs and speech acts, respectively, by HFh. Lines (5) and (6) are speech and verb acts by Buster. Though there are many standard verbs, such as the use of the verb **comfort** in line (3) above, the variety is essentially unlimited, as players have the ability to create their own verbs.

The rooms and objects in LambdaMOO are created by users themselves, who devise descriptions, and control access by other users. Users can also create objects with verbs that can be invoked by other players. As last count, the database contains 118,154 objects, including 4836 active user accounts. LambdaMOO's long existence and its user-created nature combine to give it one of the strongest senses of virtual community in the on-line world. Many users have interacted extensively with each other over many years, and users are widely acknowledged for their contribution of interesting objects. LambdaMOO is an attractive environment for experiments in AI (Foner1997; Mauldin1994), including learning. The population is generally curious and technically savvy, and users are interested in automated objects meant to display some form of intelligence.

## 3 Cobot

Cobot is a software agent residing in LambdaMOO. Like a human user, he connects via telnet, and from the point of view of the LambdaMOO server, is a user with all the rights and responsibilities implied. Once actually connected, Cobot wanders into the Living Room, where he spends most of his time. The Living Room is a central public place, frequented both by many regulars, and by users new to LambdaMOO. There are several permanent objects in the Living Room, including a couch with various features and a cuckoo clock. The Living Room usually has between five and twenty users, and is perpetually busy. Over a year, Cobot noted over 2.5 million separate events (about one event every eleven seconds)

Previously, we implemented a variety of functionality on Cobot centering around gathering and reporting *social statistics*. Cobot notes who takes what actions, and on whom. Cobot can answer queries about these statistics, and describe the similarities and differences between users. He also has a rudimentary chatting ability based on the application of information retrieval methods to large documents. He can also search the web to answer specific questions posed to him. A more complete description of Cobot's abilities, and his early experiences as a social agent in LambdaMOO, can be found in (Isbell et al.2000).

Our focus here is to make Cobot *proactive—i.e.*, let him take actions under his own initiative—in a way that is useful, interesting, or pleasing to LambdaMOO users. It is impossible to program rules anticipating when any given action is appropriate in such a complex and dynamic environment, so we applied reinforcement learning to adapt directly from user feedback. We emphasize that *Cobot's original reactive functionality remained on during the RL experiment*. Cobot's persona is largely due to this original functionality, and we felt it was most interesting, and even necessary, to add RL work in this context.

| Null Action | Choose to remain silent for this time period. |
|---|---|
| Topic Change (4) | Introduce a conversational topic. Cobot declares that he wants to discuss sports or politics, or he utters a sentence from either the sports section or political section of the Boston Globe. |
| Roll Call (2) | Initiate a "roll call," a common word play routine in LambdaMOO. For example, someone who is tired of Monica Lewinsky may emote "TIRED OF LEWINSKY ROLL CALL." Sympathetic users agree with the roll call. Cobot takes a recent utterance, and extracts either a single noun, or a verb phrase. |
| Social Commentary | Make a comment describing the current social state of the Living Room, such as "It sure is quiet" or "Everyone here is friendly." These statements are based on Cobot's statistics from recent activity. Several different utterances possible, but they are treated as a single action for RL purposes. |
| Introductions | Introduce two users who have not yet interacted in front of Cobot. |

Table 1: **The 9 RL actions available to Cobot.**

# 4 RL Background

In RL, problems of decision-making by agents interacting with uncertain environments are usually modeled as Markov decision processes (MDPs). In the MDP framework, at each time step the agent senses the state of the environment, and chooses and executes an action from the set of actions available to it in that state. The agent's action (and perhaps other uncontrolled external events) cause a stochastic change in the state of the environment. The agent receives a (possibly zero) scalar reward from the environment. The agent's goal is to choose actions so as to maximize the expected sum of rewards over some time horizon. An optimal policy is a mapping from states to actions that achieves the agent's goal.

Many RL algorithms have been developed for learning good approximations to an optimal policy from the agent's experience in its environment. At a high level, most algorithms use this experience to learn *value functions* (or *Q-values*) that map state-action pairs to the maximal expected sum of reward that can be achieved starting from that state-action pair. The learned value function is used to choose actions stochastically, so that in each state, actions with higher value are chosen with higher probability. In addition, many RL algorithms use some form of *function approximation* (parametric representations of complex value functions) both to map state-action features to their values and to map states to distributions over actions (i.e., the policy). See (Sutton and Barto1998) for an extensive introduction to RL.

In the next sections, we describe the Cobot's actions, our choice of state features, and how we dealt with multiple sources of reward. The particular RL algorithm we use is a variant of (Sutton et al.1999)'s policy gradient algorithm. Its details are beyond the scope of this paper; however, see (Shelton2000) for details. One aspect of our RL algorithm that is relevant to understanding our results is that we use a *linear function approximator* to store our policy. In other words, for each state feature, we maintain a vector of real-valued *weights* indexed by the possible actions. A positive weight for some action means that the feature increases the probability of taking that action, while a negative weight decreases the probability. The weight's magnitude determines the strength of this contribution.

# 5 Cobot's RL Actions

To have any hope of learning to behave in a way interesting to LambdaMOO users, Cobot's actions must "make sense" to them, fit in with the social chat-based environment, and minimize the risk of causing irritation. Conversation, word play, and emoting routines are among the most common activity in LambdaMOO, so we designed a set of actions along these lines, as detailed in Table 1. Many of these actions extract an utterance from the recent conversations, or from a continually changing external source, such as the online Boston Globe. Thus a single action may cause an infinite variety of behavior by Cobot.

At set time intervals (only every few minutes on average, to minimize spam), Cobot selects an action to perform from this set according to a distribution determined by the Q-values in his current state. Any rewards or punishments received before the next RL action are attributed to the current action, and used to update Cobot's value functions. It is worth remembering that Cobot has two different categories of action: those actions taken proactively as a result of the RL, and those actions taken in response to a user's action towards Cobot. Some users are certainly aware of the distinction and can easily determine which

actions fall into which category, but other users may occasionally reward or punish Cobot in response to a reactive action. Such "erroneous" rewards and punishments act as a source of noise in the training process.

## 6 The RL Reward Function

Cobot learns to behave directly from the feedback of LambdaMOO users, any of whom can reward or punish him. There are both *explicit* and *implicit* feedback mechanisms. We implemented explicit **reward** and **punish** verbs on Cobot that LambdaMOO users can invoke at any time. These verbs give a numerical (positive and negative, respectively) training signal to Cobot that is the basis of the RL. The signal is attributed as immediate feedback for the current state and RL action, and "backed up" to previous states and actions in accordance with the standard RL algorithms.

There are several standard LambdaMOO verbs that are commonly used to express, sometimes playfully, approval or disapproval. Examples of the former include the verb **hug**, and of the latter the verb **spank**. In the interest of allowing the RL process to integrate naturally with the LambdaMOO environment, we chose to accept a number of such verbs as implicit reward and punishment signals for Cobot; however, such implicit feedback is numerically weaker than the feedback generated by the explicit mechanisms.

One fundamental design choice is whether to learn a single value function for the entire community, or to learn separate value functions for each user based on individual feedback, combining the value functions of those present to determine how to act at each moment. We opted for the latter for three primary reasons.

First, it was clear that for learning to have any hope of success, ths system must represent who is present at any given moment—different users simply have different personalities and preferences. We felt that representing which users are present as additional state features would throw away valuable domain information, as the RL would have to discover on its own the primacy of user identity. Having separate reward functions for each user is thus a way of asserting the importance of identity to the learning process.

Second, despite the extremely limited number of training examples available in this domain ($< 700$ per *month*), learning must be quick and significant. Without a clear sense that their training has some impact on Cobot's behavior, users will quickly lose interest in providing feedback. A known challenge for RL is the "curse of dimensionality," (*i.e.* the size of the state space increases exponentially with the number of state features). By avoiding the need to represent the presence or absence of roughly 250 users, we are able to maintain a fairly small state space and so speed up learning.

Third, we (correctly) anticipated the fact that certain users would provide an inordinate amount of training to Cobot, and we did not want the overall policy followed by Cobot to be dominated by the preferences of these individuals. By learning separate policies for each user, and then combining these policies among those users present, we can limit the impact any single user can have on Cobot's actions.

## 7 Cobot's RL State Features

The decision to maintain and learn separate value functions for each user means that we can maintain separate state spaces as well, in the hopes of simplifying states and speeding learning. Cobot can be viewed as running a large number of separate RL processes in parallel, with each process having a different state space. The state space for a user contains a number of features containing statistics about that particular user.

LambdaMOO is a social environment, and Cobot is learning to take social actions, so we felt that his state features should contain information allowing him to gauge social activity and relationships. Table 2 provides a description of the state features used for RL by Cobot for each user. Even though we have simplified the state space by partitioning by user, the state space for a single user remains sufficiently complex to preclude standard table-based representation of value functions (also, each user's state space is effectively infinite, as there are real-valued state features). Thus, linear function approximation is used for each user's policy. Cobot's RL actions are then chosen according to a mixture of the policies of the users present. We refer the reader to (Shelton2000) for more details on the method by which policies are learned and combined.

| | |
|---|---|
| **Social Summary Vector** | A vector of four numbers: the rate at which the user is producing events; the rate at which events are being produced that are directed at the user; the percentage of the other users present who are among this user's ten most frequently interacted-with users ("playmates"); and the percentage of the other users present for whom this user is among their top ten playmates. |
| **Mood Vector** | A vector measuring the recent use of eight groups of common verbs (*e.g.*, one group includes verbs *grin* and *smile*). Verbs were grouped according to how well their usage was correlated. |
| **Rates Vector** | A vector measuring the rate at which events are produced by those present. |
| **Current Room** | The room where Cobot currently resides. |
| **Roll Call Vector** | Indicates if Cobot's currently saved roll call text has been used before, if someone has done a roll call since the last time Cobot did, and if there has been a roll call since the last time Cobot grabbed new text. |
| **Bias** | Each user has one feature that is always "on"; that is, this bias is always set to a value of 1. Intuitively, it is the feature indicating the user's "presence." |

Table 2: **State space of Cobot.** Each user has his own state space and value function; the table thus describes the state space maintained for a generic user.

# 8    Experimental Procedure and Findings

Cobot has been present in LambdaMOO more or less continuously since September, 1999. The RL version of Cobot debuted May 10, 2000. Again, Cobot's various reactive functionality was left intact for the duration of the RL experiment. Cobot is a working system with real human users, and we wanted to perform the RL experiment in this context. Upon launching the RL functionality publicly in the Living Room, Cobot logged all RL-related data (states visited, actions taken, rewards received from each user, parameters of the value functions, etc.) from May 10 until October 10, 2000. During this time, 63123 RL actions were taken (in addition, of course, to many more reactive non-RL actions), and 3171 reward and punishment events were received from 254 different users. The findings we now summarize are based on these extensive logs:

**Inappropriateness of average reward.** The most standard and obvious sign of successful RL would be an increase in the average reward over time. Instead, as shown in Figure 1a, the average cumulative reward received by Cobot actually goes *down*. However, rather than indicating that users are becoming more dissatisfied as Cobot learns, the decay in reward reveals some peculiarities of human feedback in such an open-ended environment. There are at least two difficulties with average cumulative reward in an environment of human users. The first is that humans are fickle, and their tastes and preferences may drift over time. Indeed, our experiences as users, and with the original reactive functionality of Cobot, suggest that novelty is highly valued in LambdaMOO. Thus a feature that is popular and exciting to users when it is introduced may eventually become an irritant (there are many examples of this phenomenon). In RL terminology, we do not have a fixed, consistent reward function, and thus we are always learning a moving target. While difficult to quantify in such a complex environment, this phenomenon is sufficiently prevalent in LambdaMOO to cast serious doubts on the use of average cumulative reward as the primary measure of performance.

The second and related difficulty is that even when users do maintain relatively fixed preferences, they tend to give Cobot less feedback of either type (reward or punishment) as he manages to learn their preferences accurately. Simply put, once Cobot seems to be behaving as they wish, users feel no need to continually provide reward for his "correct" actions or to punish him for the occasional "mistake." This reward pattern is in contrast to typical RL applications, where there is an automated and indefatigable reward source. Strong empirical evidence for this second phenomenon is provided by User M and User S. These two users were among Cobot's most dedicated trainers, each had strong preferences for certain actions, and Cobot learned to strongly modify his behavior in their presence to match their preferences. Nevertheless, both users tended to provide less frequent feedback to Cobot as the experiment progressed, as shown in Figure 1a. We conclude that there are serious conceptual difficulties with the use of average cumulative reward in such a human-centric application of RL, and that alternative measures must be investigated, which we do below.

**A small set of dedicated "parents."** Among the 254 users who gave at least one reward or punishment event to Cobot, 218 gave 20 or fewer, while 15 gave 50 or more. Thus, we found that while many users exhibited a passing interest in training Cobot, there was a small group that was willing to invest nontrivial time and effort in teaching Cobot their preferences. In particular, User M and User S, generated 594 and 69 rewards and punishments events, respectively.[3]

| User O | 40° | **Roll Call**. User O appears to especially dislike roll call actions when there have been repeated roll calls and/or Cobot is repeating the same roll calls. |
| | 27 – 33° | **Rates**. The overall rate of events being generating has slightly more relevance than that of the rate of events being generated just by User O. |
| User B | 10° | **Social Summary**. User B is effected by the presence of his friends. Not shown here are other Social Summary features (deviating about 6 degrees). It appears that User B is more likely to ignore Cobot when he is with many friends. |
| User C | 34° | **Roll Call**. User C appears to have strong preferences about Cobot's behavior when a "roll call party" is in progress (*i.e.*, everyone is generating roll calls). |
| User P | 22° | **Room**. User P would follow Cobot to his home, where he is generally alone, and has trained him there. He appears to have different preferences for Cobot under those circumstances. |

Table 3: **Relevant features for users with non-uniform policies.** Several of our top users had some features that deviated from their bias feature. The second column indicates the number of degrees between the weight vectors for those features and the weight vectors for the bias feature. We have only included features that deviated by more than 10 degrees. For the users above the double line, we have included only features whose weights had a length greater than 0.2. Each of these users had bias weights of length greater than 1. For those below the line, we have included only features with a length greater than 0.1 (these all had bias weights of length much less than 1).

**Some parents have strong opinions.** For the vast majority of users who participated in the RL training of Cobot, the policy learned was quite close to the uniform distribution. Quantification of this statement is somewhat complex, since policies are dependent on state. However, we observed that for most users the learned policy's dependence on state was weak, and the resulting distribution near uniform (though there are interesting and notable exceptions, as we shall see below). This result is perhaps to be expected: most users provided too little feedback for Cobot to detect strong preferences, and may not have been exhibiting strong and consistent preferences in the feedback they did provide. However, there was again a small group of users for whom a highly non-uniform policy was learned. In particular, for Users M and S mentioned above, the resulting policies were relatively independent of state and their entropies were 0.03 and 1.93, respectively. (The entropy of the uniform distribution over the actions is 2.2.) Several other users also exhibited less dramatic but still non-uniform distributions. User M seemed to have a strong preference for roll call actions, with the learned policy selecting these with probability 0.99, while User S preferred social commentary actions, with the learned policy giving them probability 0.38. (Each action in the uniform distribution is given weight $1/9 = 0.11$.)

**Cobot learns matching policies.** In Figure 1b, we demonstrate that the policy learned by Cobot for User M does in fact reflect the empirical pattern of rewards received over time. Similar results obtain for User S, not shown here. Thus, repeated feedback given to Cobot for a non-uniform set of preferences clearly pays off in a corresponding policy.

**Cobot responds to his dedicated parents.** The policies learned by Cobot for users can have strong impact on the empirical distribution of actions he actually ends up taking in LambdaMOO. For User M, we find that his presence causes a significant shift towards his preferences. In other words, Cobot does his best to "please" these dedicated trainers whenever they arrive in the Living Room, and returns to a more uniform policy upon their departure.

**Some preferences depend on state.** Finally, we show that the policies learned by Cobot sometimes depend upon the features Cobot maintains in his state. We use two facts about the RL weights (described in Section 4) maintained by Cobot to determine which features are relevant for a given user. First, we note that by construction, the RL weights learned for the bias feature described in Table 2 represent the user's preferences *independent* of state (since this feature is always on whenever the user is present). Second, we note that because we initialized all weights to 0, only features with non-zero weights will contribute to the policy that Cobot uses. Thus, we can determine that a feature is relevant for a user if that feature's weight vector is far from that user's bias feature weight vector, *and* from the all-zero vector. For our purposes, we have used (1) the normalized inner product (the cosine of the angle between two vectors) as a measure of a feature's distance from the bias feature, and (2) a feature's weight vector length to determine if it is away from zero. These measures show that for most users, Cobot learned a state-independent policy (*e.g.*, User M prefers roll calls); however, as we can see in Table 3, Cobot has learned a policy for some users that depends upon state.

numerical reward received may be larger or smaller than 1 at any time, as implicit rewards provide fractional reward, and the user may repeatedly reward or punish an action, with the feedback being summed. For example, the total absolute value of rewards and punishments provided by User M was 607.63 over 594 feedback events.

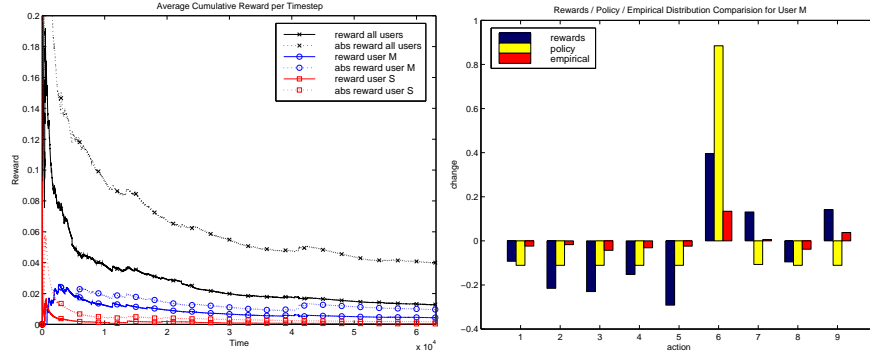

Figure 1: **a) Average Cumulative Reward Over Time. b) Rewards received, policy learned, and effect on actions for User M.** Figure **a)** shows that average cumulative reward decreases over time, for both total and absolute reward; however, Figure **b** shows that proper learning is taking place. For each of the RL actions, three quantities are shown. The blue bars (left) show the average reward given by User M for each action (the average reward given by User M across all actions has been subtracted off to indicate relative preferences). The yellow bars (middle) show the policy learned by Cobot for User M (the probability assigned to each action in the uniform distribution (1/9) has been subtracted off). The red bars (right) show the empirical frequency with which each action was taken in the presence of User M (minus the empirical frequency with which that action was taken by Cobot over all time steps). These bars indicate the extent to which the presence of User M biases Cobot's behavior towards M's preferences. We see that the policy learned by Cobot for User M aligns nicely with the preferences expressed by M and that Cobot's behavior shifts strongly towards the learned policy for User M whenever M is present. To go beyond a qualitative visual analysis, we have defined a metric that measures the extent to which two rankings of actions agree, while taking into account that some actions are extremely close in the each ranking. The details are beyond the scope of the paper, but the agreement between the action rankings shown here are in near-perfect agreement by this measure. Similar results obtain for User S.

## 9 Conclusions

We have reported on our efforts to apply reinforcement learning in a complex human online social environment where many of the standard assumptions (stationary rewards, Markovian behavior, appropriateness of average reward) are clearly violated. We feel that the results obtained with Cobot so far are compelling, and offer promise for the application of RL in such open-ended social settings. Cobot continues to take RL actions and receive rewards and punishments from LambdaMOO users, and we plan to continue and embellish this work as part of our overall efforts on Cobot.

## Footnotes

[1] Here we mean "responding only to human-invoked interaction", rather than "non-deliberative".

[2] Characters in LambdaMOO have gender. Cobot's description to users indicates that he is male.

[3]By "event", we simply mean an RL action that received some feedback. The actual absolute

## References

Foner, L. (1997). Entertaining Agents: a Sociological Case Study. In *Proceedings of the First International Conference on Autonomous Agents*.

Isbell, C. L., Kearns, M., Kormann, D., Singh, S., and Stone, P. (2000). Cobot in LambdaMOO: A Social Statistics Agent. *To appear in Proceedings of AAAI-2000*.

Mauldin, M. (1994). Chatterbots, TinyMUDs, and the Turing Test: Entering the Loebner Prize Competition. In *Proceedings of the Twelfth National Conference on Artificial Intelligence*.

Shelton, C. R. (2000). Balancing Multiple Sources of Reward in Reinforcement Learning. *Submitted for publication in Neural Information Processing Systems-2000*.

Singh, S., Kearns, M., Littman, D., and Walker, M. (2000). Empirical Evaluation of a Reinforcement Learning Dialogue System. *To appear in Proceedings of AAAI-2000*.

Sutton, R. S. and Barto, A. G. (1998). *Reinforcement Learning: An Introduction*. MIT Press, Cambridge, MA.

Sutton, R. S., McAllester, D., Singh, S., and Mansour, Y. (1999). Policy gradient methods for reinforcement learning with function approximation. *In Neural Information Processing Systems-1999*.
